# Graph Matching for Shape Retrieval

**Benoit Huet, Andrew D.J. Cross and Edwin R. Hancock**\*
Department of Computer Science, University of York
York, Y01 5DD, UK

## Abstract

This paper describes a Bayesian graph matching algorithm for data-mining from large structural data-bases. The matching algorithm uses edge-consistency and node attribute similarity to determine the *a posteriori* probability of a query graph for each of the candidate matches in the data-base. The node feature-vectors are constructed by computing normalised histograms of pairwise geometric attributes. Attribute similarity is assessed by computing the Bhattacharyya distance between the histograms. Recognition is realised by selecting the candidate from the data-base which has the largest *a posteriori* probability. We illustrate the recognition technique on a data-base containing 2500 line patterns extracted from real-world imagery. Here the recognition technique is shown to significantly outperform a number of algorithm alternatives.

## 1 Introduction

Since Barrow and Popplestone [1] first suggested that relational structures could be used to represent and interpret 2D scenes, there has been considerable interest in the machine vision literature in developing practical graph-matching algorithms [8, 3, 10]. The main computational issues are how to compare relational descriptions when there is significant structural corruption [8, 10] and how to search for the best match [3]. Despite resulting in significant improvements in the available methodology for graph-matching, there has been little progress in applying the resulting algorithms to large-scale object recognition problems. Most of the algorithms developed in the literature are evaluated for the relatively simple problem of matching a model-graph against a scene known to contain the relevant structure. A more realistic problem is that of taking a large number (maybe thousands) of scenes and retrieving the ones that best match the model. Although this problem is key to data-mining from large libraries of visual information, it has invariably been approached using low-level feature comparison techniques. Very little effort [7, 4] has been devoted to matching

higher-level structural primitives such as lines, curves or regions. Moreover, because of the perceived fragility of the graph matching process, there has been even less effort directed at attempting to retrieve shapes using relational information.

Here we aim to fill this gap in the literature by using graph-matching as a means of retrieving the shape from a large data-based that most closely resembles a query shape. Although the indexation images in large data-bases is a problem of current topicality in the computer vision literature [5, 6, 9], the work presented in this paper is more ambitious. Firstly, we adopt a structural abstraction of the shape recognition problem and match using attributed relational graphs. Each shape in our data-base is a pattern of line-segments. The structural abstraction is a nearest neighbour graph for the centre-points of the line-segments. In addition, we exploit attribute information for the line patterns. Here the geometric arrangement of the line-segments is encapsulated using a histogram of Euclidean invariant pairwise (binary) attributes. For each line-segment in turn we construct a normalised histogram of relative angle and length with the remaining line-segments in the pattern. These histograms capture the global geometric context of each line-segment. Moreover, we interpret the pairwise geometric histograms as measurement densities for the line-segments which we compare using the Bhattacharyya distance.

Once we have established the pattern representation, we realise object recognition using a Bayesian graph-matching algorithm. This is a two-step process. Firstly, we establish correspondence matches between the individual tokens in the query pattern and each of the patterns in the data-base. The correspondences matches are sought so as to maximise the *a posteriori* measurement probability. Once the MAP correspondence matches have been established, then the second step in our recognition architecture involves selecting the line-pattern from the data-base which has maximum matching probability.

## 2 MAP Framework

Formally our recognition problem is posed as follows. Each ARG in the database is a triple, $G = (V_G, E_G, A_G)$, where $V_G$ is the set of vertices (nodes), $E_G$ is the edge set $(E_G \subset V_G \times V_G)$, and $A_G$ is the set of node attributes. In our experimental example, the nodes represent line-structures segmented from 2D images. The edges are established by computing the N-nearest neighbour graph for the line-centres. Each node $j \in V_G$ is characterised by a vector of attributes, $\underline{x}_j$ and hence $A_G = \{\underline{x}_j | j \in V_G\}$. In the work reported here the attribute-vector is represents the contents of a normalised pairwise attribute histogram.

The data-base of line-patterns is represented by the set of ARG's $\mathcal{D} = \{G\}$. The goal is to retrieve from the data-base $\mathcal{D}$, the individual ARG that most closely resembles a query pattern $Q = (V_Q, E_Q, A_Q)$. We pose the retrieval process as one of associating with the query the graph from the data-base that has the largest *a posteriori* probability. In other words, the class identity of the graph which most closely corresponds to the query is

$$\omega_Q = \arg \max_{G' \in \mathcal{D}} P(G'|Q)$$

However, since we wish to make a detailed structural comparison of the graphs, rather than comparing their overall statistical properties, we must first establish a set of best-match correspondences between each ARG in the data-base and the query $Q$. The set of correspondences between the query $Q$ and the ARG $G$ is a relation $f_G : V_G \mapsto V_Q$ over the vertex sets of the two graphs. The mapping function consists of a set of Cartesian pairings between the nodes of the two graphs,

i.e. $f_G = \{(a,\alpha); a \in V_G, \alpha \in V_Q\} \subseteq V_G \times V_Q$. Although this may appear to be a brute force method, it must be stressed that we view this process of correspondence matching as the final step in the filtering of the line-patterns. We provide more details of practical implementation in the experimental section of this paper.

With the correspondences to hand we can re-state our maximum *a posteriori* probability recognition objective as a two step process. For each graph $G$ in turn, we locate the maximum *a posteriori* probability mapping function $f_G$ onto the query $Q$. The second step is to perform recognition by selecting the graph whose mapping function results in the largest matching probability. These two steps are succinctly captured by the following statement of the recognition condition

$$\omega_Q = \arg \max_{G' \in \mathcal{D}} \max_{f_{G'}} P(f_{G'}|G', Q)$$

This global MAP condition is developed into a useful local update formula by applying the Bayes formula to the *a posteriori* matching probability. The simplification is as follows

$$P(f_G|G,Q) = \frac{p(A_G, A_Q|f_G)P(f_G|V_G, E_G, V_Q, E_Q)P(V_G, E_G)P(V_Q, E_Q)}{P(G)P(Q)}$$

The terms on the right-hand side of the Bayes formula convey the following meaning. The conditional measurement density $p(A_G, A_Q|f_G)$ models the measurement similarity of the node-sets of the two graphs. The conditional probability $P(f_G|E_G, E_Q)$ models the structural similarity of the two graphs under the current set of correspondence matches. The assumptions used in developing our simplification of the *a posteriori* matching probability are as follows. Firstly, we assume that the joint measurements are conditionally independent of the structure of the two graphs provided that the set of correspondences is known, i.e. $P(A_G, A_Q|f_G, E_G, V_G, E_Q, V_Q) = P(A_G, A_Q|f_G)$. Secondly, we assume that there is conditional independence of the two graphs in the absence of correspondences. In other words, $P(V_G, E_G, V_Q, E_Q) = P(V_Q, E_Q)P(V_G, E_G)$ and $P(G,Q) = P(G)P(Q)$. Finally, the graph priors $P(V_G, E_G)$, $P(V_Q, E_Q)$ $P(G)$ and $P(Q)$ are taken as uniform and are eliminated from the decision making process.

To continue our development, we first focus on the conditional measurement density, $p(A_G, A_Q|f_G)$ which models the process of comparing attribute similarity on the nodes of the two graphs. Assuming statistical independence of node attributes, the conditional measurement density $p(A_G, A_Q|f_G)$, can be factorised over the Cartesian pairs $(a, \alpha) \in V_G \times V_Q$ which constitute the the correspondence match $f_G$ in the following manner

$$p(A_G, A_Q|f_G) = \prod_{(a,\alpha) \in f_G} p(\underline{x}_a, \underline{x}_\alpha|f_G(a) = \alpha)$$

As a result the correspondence matches may be optimised using a simple node-by-node discrete relaxation procedure. The rule for updating the match assigned to the node $a$ of the graph $G$ is

$$f_G(a) = \arg \max_{\alpha \in V_Q \cup \{\Phi\}} p(\underline{x}_a, \underline{x}_\alpha)|f(a) = \alpha)P(f_G|E_G, E_Q)$$

In order to model the structural consistency of the set of assigned matches,we turn to the framework recently reported by Finch, Wilson and Hancock [2]. This work provides a framework for computing graph-matching energies using the weighted Hamming distance between matched cliques. Since we are dealing with a large-scale object recognition system, we would like to minimise the computational overheads associated with establishing correspondence matches. For this reason, rather than

working with graph neighbourhoods or cliques, we chose to work with the relational units of the smallest practical size. In other words we satisfy ourself with measuring consistency at the edge level. For edge-units, the structural matching probability $P(f_G|V_G, E_G, V_Q, E_Q)$ is computed from the formula

$$\ln P(f_G|V_G, E_G, V_G, E_Q) = \sum_{(a,b)\in E_G} \sum_{(\alpha,\beta)\in E_Q} \{\ln(1-P_e)s_{a,\alpha}s_{b,\beta}+\ln P_e(1-s_{a,\alpha}s_{b,\beta})\}$$

where $P_e$ is the probability of an error appearing on one of the edges of the matched structure. The $s_{a,\alpha}$ are assignment variables which are used to represent the current state of match and convey the following meaning

$$s_{a,\alpha} = \begin{cases} 1 & \text{if } f_G(a) = \alpha \\ 0 & \text{otherwise} \end{cases}$$

## 3   Histogram-based consistency

We now furnish some details of the shape retrieval task used in our experimental evaluation of the recognition method. In particular, we focus on the problem of recognising 2D line patterns in a manner which is invariant to rotation, translation and scale. The raw information available for each line segment are its orientation (angle with respect to the horizontal axis) and its length (see figure 1). To illustrate how the Euclidean invariant pairwise feature attributes are computed, suppose that we denote the line segments associated with the nodes indexed $a$ and $b$ by the vectors $\underline{v}_a$ and $\underline{v}_b$ respectively. The vectors are directed away from their point of intersection. The pairwise relative angle attribute is given by

$$\theta_{a,b} = \arccos\left[\frac{\underline{v}_a \cdot \underline{v}_b}{|\underline{v}_a||\underline{v}_b|}\right]$$

From the relative angle we compute the directed relative angle. This involves giving

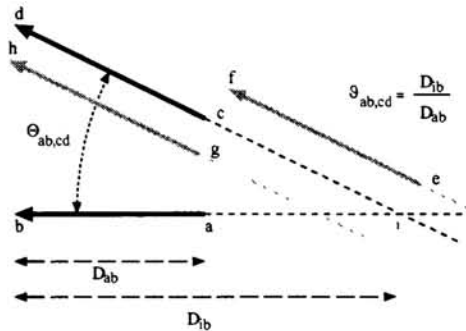

Figure 1: Geometry for shape representation

the relative angle a positive sign if the direction of the angle from the baseline $\underline{v}_a$ to its partner $\underline{v}_b$ is clockwise and a negative sign if it is counter-clockwise. This allows us to extend the range of angles describing pairs of segments from $[0,\pi]$ to $[-\pi,\pi]$.

The directed relative position $\vartheta_{a,b}$ is represented by the normalised length ratio between the oriented baseline vector $\underline{v}_a$ and the vector $\underline{v}\prime$ joining the end ($b$) of the baseline segment ($ab$) to the intersection of the segment pair ($cd$).

$$\vartheta_{a,b} = \frac{1}{\frac{1}{2} + \frac{D_{ib}}{D_{ab}}}$$

The physical range of this attribute is $(0,1]$. A relative position of $0$ indicates that the two segments are parallel, while a relative position of $1$ indicates that the two segments intersect at the middle point of the baseline.

The Euclidean invariant angle and position attributes $\theta_{a,b}$ and $\vartheta_{a,b}$ are binned in a histogram. Suppose that $S_a(\mu,\nu) = \{(a,b)|\theta_{a,b} \in A_\mu \wedge \vartheta_{a,b} \in R_\nu \wedge b \in V_D\}$ is the set of nodes whose pairwise geometric attributes with the node $a$ are spanned by the range of directed relative angles $A_\mu$ and the relative position attribute range $R_\nu$. The contents of the histogram bin spanning the two attribute ranges is given by $H_a(\mu,\nu) = |S_a(\mu,\nu)|$. Each histogram contains $n_A$ relative angle bins and $n_R$ length ratio bins. The normalised geometric histogram bin-entries are computed as follows

$$h_a(\mu,\nu) = \frac{H_a(\mu,\nu)}{\sum_{\mu'=1}^{n_A}\sum_{\nu'=1}^{n_R}H_a(\mu,\nu)}$$

The probability of match between the pattern-vectors is computed using the Bhattacharyya distance between the normalised histograms.

$$P(f(a)=\alpha|\underline{x}_a,\underline{x}_\alpha) = \frac{\sum_{\mu=1}^{n_A}\sum_{\nu=1}^{n_R}h_a(\mu,\nu)h_\alpha(\mu,\nu)}{\sum_{j'\in Q}\sum_{\mu'=1}^{n_A}\sum_{\nu'=1}^{n_R}h_a(\mu,\nu)h_\alpha(\mu,\nu)} = \exp[-B_{a,\alpha}]$$

With this modelling ingredient, the condition for recognition is

$$\omega_Q = \arg\min_{G'\in\mathcal{D}}\sum_{(a,b)\in E'_G}\sum_{(\alpha,\beta)\in E_Q}\left\{-B_{a,\alpha}-B_{b,\beta}+\ln(1-P_e)s_{a,\alpha}s_{b,\beta}+\ln P_e(1-s_{a,\alpha}s_{b,\beta})\right\}$$

## 4  Experiments

The aim in this section is to evaluate the graph-based recognition scheme on a database of real-world line-patterns. We have conducted our recognition experiments with a data-base of 2500 line-patterns each containing over a hundred lines. The line-patterns have been obtained by applying line/edge detection algorithms to the raw grey-scale images followed by polygonisation. For each line-pattern in the database, we construct the six-nearest neighbour graph. The feature extraction process together with other details of the data used in our study are described in recent papers where we have focussed on the issues of histogram representation [4] and the optimal choice of the relational structure for the purposes of recognition. In order to prune the set of line-patterns for detailed graph-matching we select about 10% of the data-base using a two-step process. This consists of first refining the data-base using a global histogram of pairwise attributes [4]. The top quartile of matches selected in this way are then further refined using a variant of the Haussdorff distance to select the set of pairwise attributes that best match against the query.

The recognition task is posed as one of recovering the line-pattern which most closely resembles a digital map. The original images from which our line-patterns have been obtained are from a number of diverse sources. However, a subset of the images are aerial infra-red line-scan views of southern England. Two of these infra-red images correspond to different views of the area covered by the digital map. These views are obtained when the line-scan device is flying at different altitudes. The line-scan device used to obtain the aerial images introduces severe barrel distortions and hence the map and aerial images are not simply related via a Euclidean or affine transformation. The remaining line-patterns in the data-base have been extracted from trademarks and logos. It is important to stress that although the raw images are obtained from different sources, there is nothing salient about their associated line-pattern representations that allows us to distinguish them from one-another.

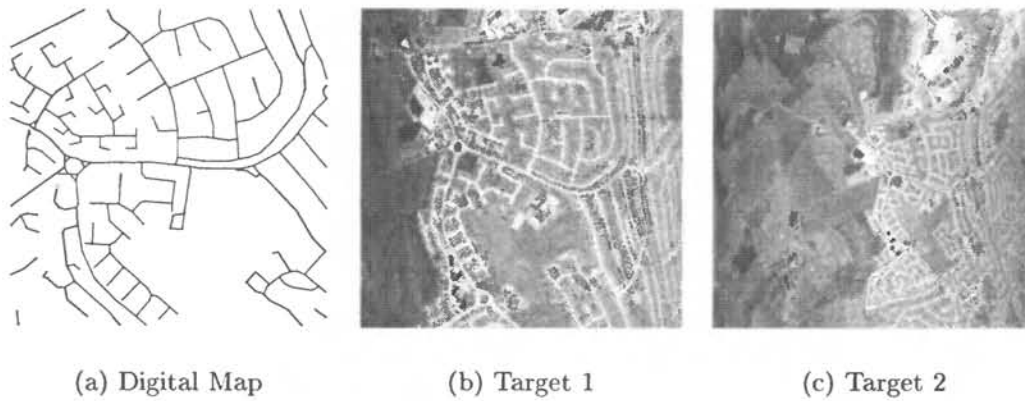

(a) Digital Map          (b) Target 1          (c) Target 2

Figure 2: Images from the data-base

Moreover, since it is derived from a digital map rather than one of the images in the data-base, the query is not identical to any of the line-patterns in the model library.

We aim to assess the importance of different attributes representation on the retrieval process. To this end, we compare node-based and the histogram-based attribute representation. We also consider the effect of taking the relative angle and relative position attributes both singly and in tandem. The final aspect of the comparison is to consider the effects of using the attributes purely for initialisation purposes and also in a persistent way during the iteration of the matching process. To this end we consider the following variants of our algorithm.

- **Non-Persistent Attributes:** Here we ignore the attribute information provided by the node-histograms after the first iteration and attempt to maximise the structural congruence of the graphs.

- **Local attributes:** Here we use only the single node attributes rather than an attribute histogram to model the *a posteriori* matching probabilities.

| Graph Matching Strategy | Retrieval Accuracy | Iterations per recall |
|---|---|---|
| Rel. Position Attribute (Initialisation only) | 39% | 5.2 |
| Rel. Angle Attribute (Initialisation only) | 45% | 4.75 |
| Rel. Angle + Position Attributes (Initialisation only) | 58% | 4.27 |
| 1D Rel. Position Histogram (Initialisation only) | 42% | 4.7 |
| 1D Rel. Angle Histogram (Initialisation only) | 59% | 4.2 |
| 2D Histogram (Initialisation only) | 68% | 3.9 |
| Rel. Position Attribute (Persistent) | 63% | 3.96 |
| Rel. Angle Attribute (Persistent) | 89% | 3.59 |
| Rel. Angle + Position Attributes (Persistent) | 98% | 3.31 |
| 1D Rel. Position Histogram (Persistent) | 66% | 3.46 |
| 1D Rel. Angle Histogram (Persistent) | 92% | 3.23 |
| 2D Histogram (Persistent) | 100% | 3.12 |

Table 1: Recognition performance of various recognition strategies averaged over 26 queries in a database of 260 line-patterns

In Table 1 we present the recognition performance for each of the recognition strategies in turn. The table lists the recall performance together with the average number

of iterations per recall for each of the recognition strategies in turn. The main features to note from this table are as follows. Firstly, the iterative recall using the full histogram representation outperforms each of the remaining recognition methods in terms of both accuracy and computational overheads. Secondly, it is interesting to compare the effect of using the histogram in the initialisation-only and iteration persistent modes. In the latter case the recall performance is some 32% better than in the former case. In the non-persistent mode the best recognition accuracy that can be obtained is 68%. Moreover, the recall is typically achieved in only 3.12 iterations as opposed to 3.9 (average over 26 queries on a database of 260 images). Finally, the histogram representation provides better performance, and more significantly, much faster recall than the single-attribute similarity measure. When the attributes are used singly, rather than in tandem, then it is the relative angle that appears to be the most powerful.

## 5  Conclusions

We have presented a practical graph-matching algorithm for data-mining in large structural libraries. The main conclusion to be drawn from this study is that the combined use of structural and histogram information improves both recognition performance and recall speed. There are a number of ways in which the ideas presented in this paper can be extended. Firstly, we intend to explore more a perceptually meaningful representation of the line patterns, using grouping principals derived from Gestalt psychology. Secondly, we are exploring the possibility of formulating the filtering of line-patterns prior to graph matching using Bayes decision trees.

## References

[1] H. Barrow and R. Popplestone. Relational descriptions in picture processing. *Machine Intelligence*, 5:377–396, 1971.

[2] A. Finch, R. Wilson, and E. Hancock. Softening discrete relaxation. *Advances in NIPS 9, Edited by M. Mozer, M. Jordan and T. Petsche, MIT Press*, pages 438–444, 1997.

[3] S. Gold and A. Rangarajan. A graduated assignment algorithm for graph matching. *IEEE PAMI*, 18:377–388, 1996.

[4] B. Huet and E. Hancock. Relational histograms for shape indexing. *IEEE ICCV*, pages 563–569, 1998.

[5] W. Niblack *et al.*. The QBIC project: Querying images by content using color, texture and shape. *Image and Vision Storage and Retrieval*, 173–187, 1993.

[6] A. P. Pentland, R. W. Picard, and S. Scarloff. Photobook: tools for content-based manipulation of image databases. *Storage and Retrieval for Image and Video Database II*, pages 34–47, February 1994.

[7] K. Sengupta and K. Boyer. Organising large structural databases. *IEEE PAMI*, 17(4):321–332, 1995.

[8] L. Shapiro and R. Haralick. A metric for comparing relational descriptions. *IEEE PAMI*, 7(1):90–94, 1985.

[9] M. Swain and D. Ballard. Color indexing. *International Journal of Computer Vision*, 7(1):11–32, 1991.

[10] R. Wilson and E. R. Hancock. Structural matching by discrete relaxation. *IEEE PAMI*, 19(6):634–648, June 1997.
